# Operational Fault Tolerance of CMAC Networks

**Michael J. Carter**      **Franklin J. Rudolph**      **Adam J. Nucci**
Intelligent Structures Group
Department of Electrical and Computer Engineering
University of New Hampshire
Durham, NH 03824-3591

## ABSTRACT

The performance sensitivity of Albus' CMAC network was studied for the scenario in which faults are introduced into the adjustable weights after training has been accomplished. It was found that fault sensitivity was reduced with increased generalization when "loss of weight" faults were considered, but sensitivity was increased for "saturated weight" faults.

## 1    INTRODUCTION

Fault-tolerance is often cited as an inherent property of neural networks, and is thought by many to be a natural consequence of "massively parallel" computational architectures. Numerous anecdotal reports of fault-tolerance experiments, primarily in pattern classification tasks, abound in the literature. However, there has been surprisingly little rigorous investigation of the fault-tolerance properties of various network architectures in other application areas. In this paper we investigate the fault-tolerance of the CMAC (Cerebellar Model Arithmetic Computer) network [Albus 1975] in a systematic manner. CMAC networks have attracted much recent attention because of their successful application in robotic manipulator control [Ersu 1984, Miller 1986, Lane 1988]. Since fault-tolerance is a key concern in critical control tasks, there is added impetus to study

this aspect of CMAC performance. In particular, we examined the effect on network performance of faults introduced into the adjustable weight layer after training has been accomplished in a fault-free environment. The degradation of approximation error due to faults was studied for the task of learning simple real functions of a single variable. The influence of receptive field width and total CMAC memory size on the fault sensitivity of the network was evaluated by means of simulation.

## 2   THE CMAC NETWORK ARCHITECTURE

The CMAC network shown in Figure 1 implements a form of distributed table lookup. It consists of two parts: 1) an address generator module, and 2) a layer of adjustable weights. The address generator is a fixed algorithmic transformation from the input space to the space of weight addresses. This transformation has two important properties: 1) Only a fixed number C of weights are activated in response to any particular input, and more importantly, only these weights are adjusted during training; 2) It is *locally generalizing*, in the sense that any two input points separated by Euclidean distance less than some threshold produce activated weight subsets that are close in Hamming distance, i.e. the two weight subsets have many weights in common. Input points that are separated by more than the threshold distance produce non-overlapping activated weight subsets. The first property gives rise to the extremely fast training times noted by all CMAC investigators. The number of weights activated by any input is referred to as the "generalization parameter", and is typically a small number ranging from 4 to 128 in practical applications [Miller 1986]. Only the activated weights are summed to form the response to the current input. A simple delta rule adjustment procedure is used to update the activated weights in response to the presentation of an input-desired output exemplar pair. Note that there is no adjustment of the address generator transformation during learning, and indeed, there are no "weights" available for adjustment in the address generator. It should also be noted that the hash-coded mapping is in general necessary because there are many more resolution cells in the input space than there are unique finite combinations of weights in the physical memory. As a result, the local generalization property will be disturbed because some distant inputs share common weight addresses in their activated weight subsets due to hashing collisions.

While the CMAC network readily lends itself to the task of learning and mimicking multidimensional nonlinear transformations, the investigation of network fault-tolerance in this setting is daunting! For reasons discussed in the next section, we opted to study CMAC fault-tolerance for simple one-dimensional input and output spaces without the use of the hash-coded mapping.

## 3   FAULT-TOLERANCE EXPERIMENTS

We distinguish between two types of fault-tolerance in neural networks [Carter 1988]: operational fault-tolerance and learning fault-tolerance. Operational fault-tolerance deals with the sensitivity of network performance to faults introduced after learning has been

accomplished in a fault-free environment. Learning fault-tolerance refers to the sensitivity of network performance to faults (either permanent or transient) which are present during training. It should be noted that the term fault-tolerance as used here applies only to faults that represent perturbations in network parameters or topology, and does not refer to noisy or censored input data. Indeed, we believe that the latter usage is both inappropriate and inconsistent with conventional usage in the computer fault-tolerance community.

## 3.1     EXPERIMENT DESIGN PHILOSOPHY

Since the CMAC network is widely used for learning nonlinear functions (e.g. the motor drive voltage to joint angle transformation for a multiple degree-of-freedom robotic manipulator), the obvious measure of network performance is function approximation error. The sensitivity of approximation error to faults is the subject of this paper. There are several types of faults that are of concern in the CMAC architecture. Faults that occur in the address generator module may ultimately have the most severe impact on approximation error since the selection of incorrect weight addresses will likely produce a bad response. On the other hand, since the address generator is an algorithm rather than a true network of simple computational units, the fault-tolerance of any serial processor implementation of the algorithm will be difficult to study. For this reason we initially elected to study the fault sensitivity of the adjustable weight layer only.

The choice of fault types and fault placement strategies for neural network fault tolerance studies is not at all straightforward. Unlike classical fault-tolerance studies in digital systems which use "stuck-at-zero" and "stuck-at-one" faults, neural networks which use analog or mixed analog/digital implementations may suffer from a host of fault types. In order to make some progress, and to study the fault tolerance of the CMAC network at the architectural level rather than at the device level, we opted for a variation on the "stuck-at" fault model of digital systems. Since this study was concerned only with the adjustable weight layer, and since we assumed that weight storage is most likely to be digital (though this will certainly change as good analog memory technologies are developed), we considered two fault models which are admittedly severe. The first is a "loss of weight" fault which results in the selected weight being set to zero, while the second is a "saturated weight" fault which might correspond to the situation of a stuck-at-one fault in the most significant bit of a single weight register.

The question of fault placement is also problematic. In the absence of a specific circuit level implementation of the network, it is difficult to postulate a model for fault distribution. We adopted a somewhat perverse outlook in the hope of characterizing the network's fault tolerance under a worst-case fault placement strategy. The insight gained will still prove to be valuable in more benign fault placement tests (e.g. random fault placement), and in addition, if one can devise network modifications which yield good fault-tolerance in this extreme case, there is hope of still better performance in more

typical instances of circuit failure. When placing "loss of weight" faults, we attacked large magnitude weight locations first, and continued to add more such faults to locations ranked in descending order of weight magnitude. Likewise, when placing saturated weight faults we attacked small magnitude weight locations first, and successive faults were placed in locations ordered by ascending weight magnitude. Since the activated weights are simply summed to form a response in CMAC, faults of both types create an error in the response which is equal to the weight change in the faulted location. Hence, our strategy was designed to produce the maximum output error for a given number of faults. In placing faults of either type, however, we did not place two faults within a single activated weight subset. Our strategy was thus not an absolute worst-case strategy, but was still more stressful than a purely random fault placement strategy. Finally, we did not mix fault types in any single experiment.

The fault tolerance experiments presented in the next section all had the same general structure. The network under study was trained to reproduce (to a specified level of approximation error) a real function of a single variable, y=f(x), based upon presentation of (x,y) exemplar pairs. Faults of the types described previously were then introduced, and the resulting degradation in approximation error was logged versus the number of faults. Many such experiments were conducted with varying CMAC memory size and generalization parameter while learning the same exemplar function. We considered smoothly varying functions (sinusoids of varying spatial frequency) and discontinuous functions (step functions) on a bounded interval.

## 3.2   EXPERIMENT RESULTS AND DISCUSSION

In this section we present the results of experiments in which the function to be learned is held fixed, while the generalization parameter of the CMAC network to be tested is varied. The total number of weights (also referred to here as memory locations) is the same in each batch of experiments. Memory sizes of 50, 250, and 1000 were investigated, but only the results for the case N=250 are presented here. They exemplify the trends observed for all memory sizes.

Figure 2 shows the dependence of RMS (root mean square) approximation error on the number of loss-of-weight faults injected for generalization parameter values C=4, 8, 16. The task was that of reproducing a single cycle of a sinusoidal function on the input interval. Note that approximation error was diminished with increasing generalization at any fault severity level. For saturated weight faults, however, approximation error increased with increasing generalization! The reason for this contrasting behavior becomes clear upon examination of Figure 3. Observe also in Figure 2 that the increase in RMS error due to the introduction of a single fault can be as much as an order of magnitude. This is somewhat deceptive since the scale of the error is rather small (typically $10^{-3}$ or so), and so it may not seem of great consequence. However, as one may note in Figure 3, the effect of a single fault is highly localized, so RMS approximation error may be a poor choice of performance measure in selected

applications. In particular, saturated weight faults in nominally small weight magnitude locations create a large relative response error, and this may be devastating in real-time control applications. Loss-of-weight faults are more benign, and their impact may be diluted by increasing generalization. The penalty for doing so, however, is increased sensitivity to saturated weight faults because larger regions of the network mapping are affected by a single fault.

Figure 4 displays some of the results of fault-tolerance tests with a discontinuous exemplar function. Note the large variation in stored weight values necessary to reproduce the step function. When a large magnitude weight needed to form the step transition was faulted, the result was a double step (Figure 4(b)) or a shifted transition point (Figure 4(c)). The extent of the fault impact was diminished with decreasing generalization. Since pattern classification tasks are equivalent to learning a discontinuous function over the input feature space, this finding suggests that improved fault-tolerance in such tasks might be obtained by reducing the generalization parameter C. This would limit the shifting of pattern class boundaries in the presence of weight faults. Preliminary experiments, however, also showed that learning of discontinuous exemplar functions proceeded much more slowly with small values of the generalization parameter.

## 4    CONCLUSIONS AND OPEN QUESTIONS

The CMAC network is well-suited to applications that demand fast learning of unknown multidimensional, static mappings (such as those arising in nonlinear control and signal processing systems). The results of the preliminary investigations reported here suggest that the fault-tolerance of conventional CMAC networks may not be as great as one might hope on the basis of anecdotal evidence in the prior literature with other network architectures. Network fault sensitivity does not seem to be uniform, and the location of particularly sensitive weights is very much dependent on the exemplar function to be learned. Furthermore, the obvious fault-tolerance enhancement technique of increasing generalization (i.e. distributing the response computation over more weight locations) has the undesirable effect of **increasing** sensitivity to saturated weight faults. While the local generalization feature of CMAC has the desirable attribute of limiting the region of fault impact, it suggests that global approximation error measures may be misleading. A low value of RMS error degradation may in fact mask a much more severe response error over a small region of the mapping. Finally, one must be cautious in making assessments of the fault-tolerance of a fixed network on the basis of tests using a single mapping. Discontinuous exemplar functions produce stored weight distributions which are much more fault-sensitive than those associated with smoothly varying functions, and such functions are clearly of interest in  pattern classification.

Many important open questions remain concerning the fault-tolerance properties of the CMAC network. The effect of faults on the address generator module has yet to be determined. Collisions in the hash-coded mapping effectively propagate weight faults to

remote regions of the input space, and the impact of this phenomenon on overall fault-tolerance has not been assessed. Much more work is needed on the role that exemplar function smoothness plays in determining the fault-tolerance of a fixed topology network.

## Acknowledgements

The authors would like to thank Tom Miller, Fil Glanz, Gordon Kraft, and Edgar An for many helpful discussions on the CMAC network architecture. This work was supported in part by an Analog Devices Career Development Professorship and by a General Electric Foundation Young Faculty Grant awarded to M.J. Carter.

## References

J.S. Albus. (1975) "A new approach to manipulator control: the Cerebellar Model Articulation Controller (CMAC)," *Trans. ASME- J. Dynamic Syst., Meas., Contr. 97* ; 220-227.

M.J. Carter. (1988) "The illusion of fault-tolerance in neural networks for pattern recognition and signal processing," *Proc. Technical Session on Fault-Tolerant Integrated Systems*, Durham, NH: University of New Hampshire.

E. Ersu and J. Militzer. (1984) "Real-time implementation of an associative memory-based learning control scheme for non-linear multivariable processes," *Proc. 1st Measurement and Control Symposium on Applications of Multivariable Systems Techniques;* 109-119.

S. Lane, D. Handelman, and J. Gelfand. (1988) "A neural network computational map approach to reflexive motor control," *Proc. IEEE Intelligent Control Conf.*, Arlington, VA.

W.T. Miller. (1986) "A nonlinear learning controller for robotic manipulators," *Proc. SPIE: Intelligent Robots and Computer Vision* 726; 416-423.

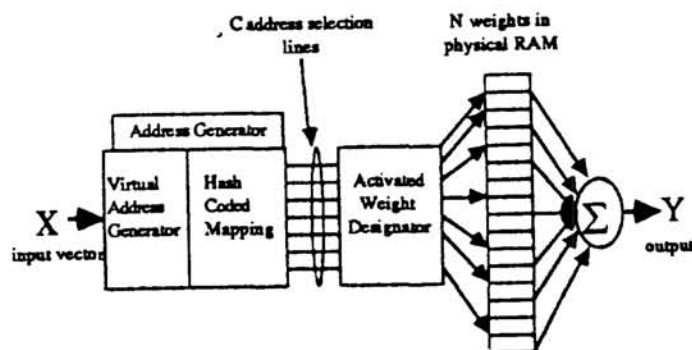

Figure 1: CMAC Network Architecture

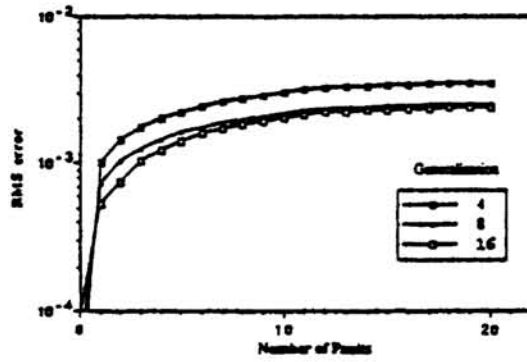

**Figure 2:** Sinusoid Approximation Error vs. Number of "Loss-of-Weight" Faults

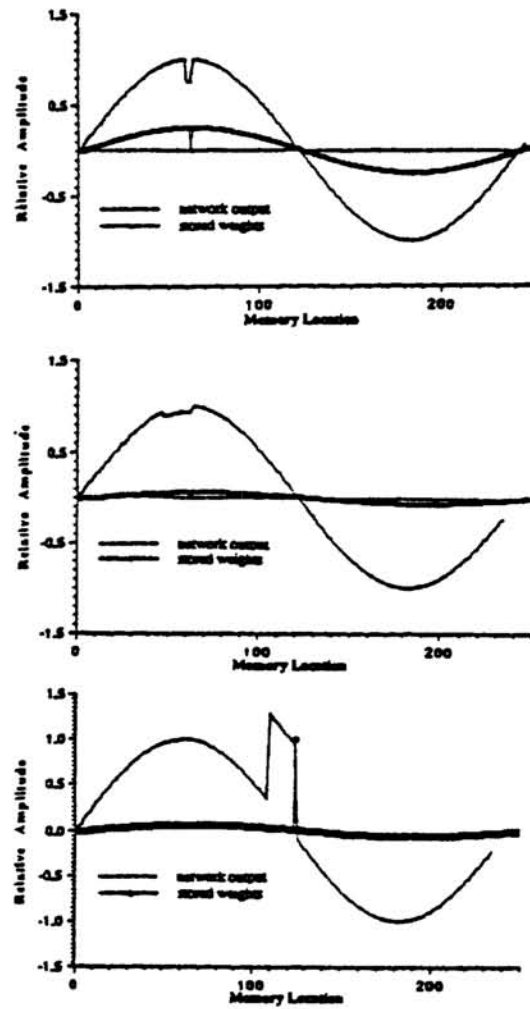

**Figure 3:** Network Response and Stored Weight Values. a) single lost weight, generalization C=4; b) single lost weight, C=16; c) single saturated weight, C=16.

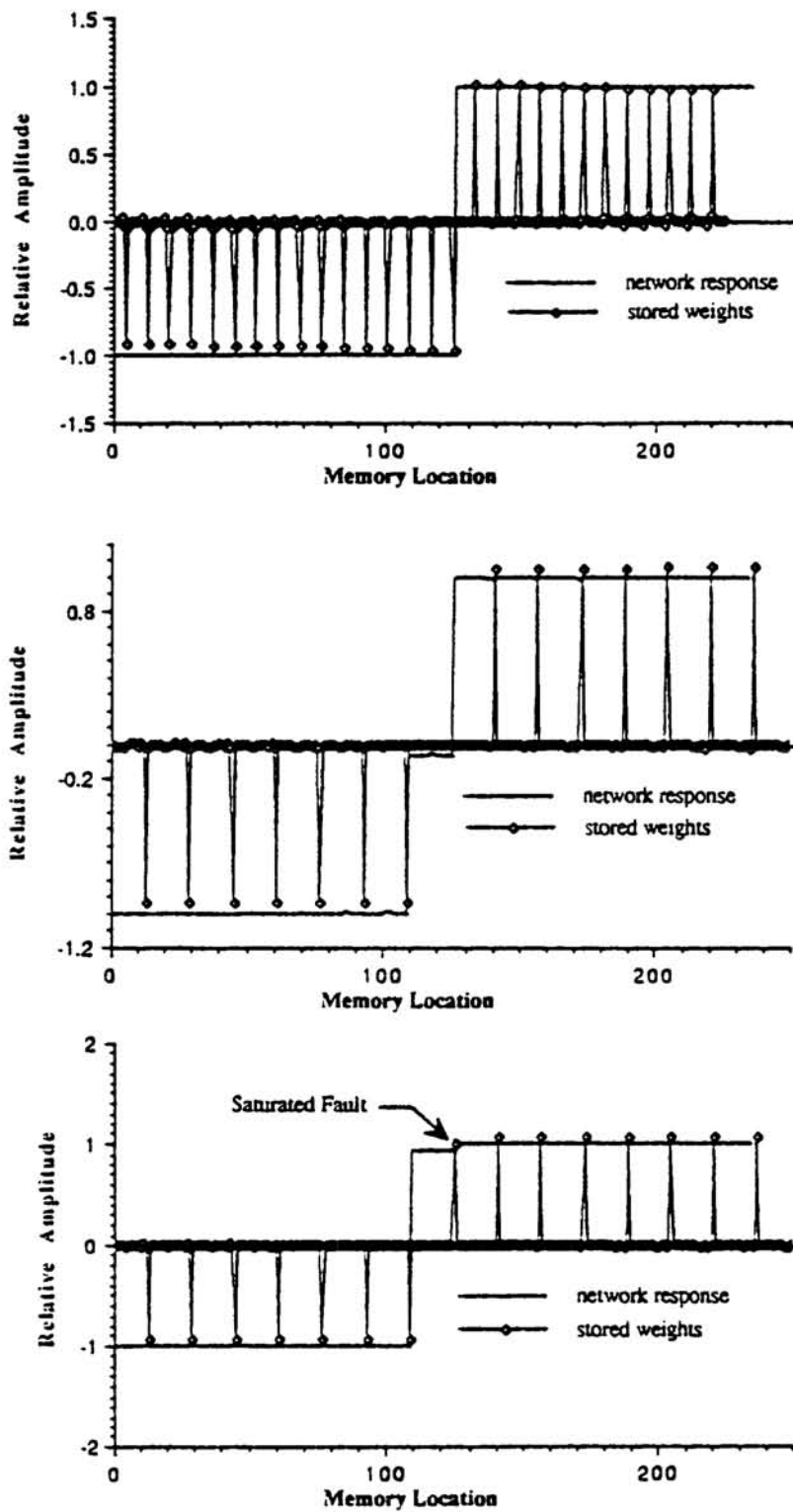

**Figure 4:** Network Response and Stored Weight Values. a) no faults, transition at location 125, C=8; b) single lost weight, C=16; c) single saturated weight, C=16.